# Walk-Sum Interpretation and Analysis of Gaussian Belief Propagation

**Jason K. Johnson, Dmitry M. Malioutov and Alan S. Willsky**
Department of Electrical Engineering and Computer Science
Massachusetts Institute of Technology
Cambridge, MA 02139
{jasonj,dmm,willsky}@mit.edu

## Abstract

This paper presents a new framework based on walks in a graph for analysis and inference in Gaussian graphical models. The key idea is to decompose correlations between variables as a sum over all walks between those variables in the graph. The weight of each walk is given by a product of edgewise partial correlations. We provide a walk-sum interpretation of Gaussian belief propagation in trees and of the approximate method of loopy belief propagation in graphs with cycles. This perspective leads to a better understanding of Gaussian belief propagation and of its convergence in loopy graphs.

## 1 Introduction

We consider multivariate Gaussian distributions defined on graphs. The nodes of the graph denote random variables and the edges indicate statistical dependencies between variables. The family of all Gauss-Markov models defined on a graph is naturally represented in the *information form* of the Gaussian density which is parameterized by the inverse covariance matrix, i.e., the *information matrix*. This information matrix is sparse, reflecting the structure of the defining graph such that only the diagonal elements and those off-diagonal elements corresponding to edges of the graph are non-zero.

Given such a model, we consider the problem of computing the mean and variance of each variable, thereby determining the marginal densities as well as the mode. In principle, these can be obtained by inverting the information matrix, but the complexity of this computation is cubic in the number of variables. More efficient recursive calculations are possible in graphs with very sparse structure – e.g., in chains, trees and in graphs with "thin" junction trees. For these models, belief propagation (BP) or its junction tree variants efficiently compute the marginals [1]. In more complex graphs, even this approach can become computationally prohibitive. Then, approximate methods such as loopy belief propagation (LBP) provide a tractable alternative to exact inference [1, 2, 3, 4].

We develop a "walk-sum" formulation for computation of means, variances and correlations that holds in a wide class of Gauss-Markov models which we call *walk-summable*. In particular, this leads to a new interpretation of BP in trees and of LBP in general. Based on this interpretation we are able to extend the previously known sufficient conditions for con-

vergence of LBP to the class of walk-summable models (which includes all of the following: trees, attractive models, and pairwise-normalizable models). Our sufficient condition is tighter than that given in [3] as the class of diagonally-dominant models is a strict subset of the class of pairwise-normalizable models. Our results also explain why no examples were found in [3] where LBP did not converge. The reason is that they presume a pairwise-normalizable model. We also explain why, in walk-summable models, LBP converges to the correct means but not to the correct variances (proving "walk-sum" analogs of results in [3]). In general, walk-summability is not necessary for LBP convergence. Hence, we also provide a tighter (essentially necessary) condition for convergence of LBP variances based on walk-summability of the LBP computation tree. This provides deeper insight into why LBP can fail to converge – because the LBP computation tree is not always well-posed – which suggests connections to [5]. This paper presents the key ideas and outlines proofs of the main results. A more detailed presentation will appear in a technical report [6].

## 2 Preliminaries

A Gauss-Markov model (GMM) is defined by a graph $\mathcal{G} = (V, \mathcal{E})$ with edge set $\mathcal{E} \subset \binom{V}{2}$, i.e., some set of two-element subsets of $V$, and a collection of random variables $x = (x_i, i \in V)$ with probability density given in *information form*[1]:

$$p(x) \propto \exp\{-\frac{1}{2}x'Jx + h'x\} \tag{1}$$

where $J$ is a symmetric positive definite ($J \succ 0$) matrix which is sparse so as to respect the graph $\mathcal{G}$: if $\{i,j\} \notin \mathcal{E}$ then $J_{i,j} = 0$. We call $J$ the *information matrix* and $h$ the *potential vector*. Let $N(i) = \{j | \{i,j\} \in \mathcal{E}\}$ denote the *neighbors* of $i$ in the graph. The mean $\mu \equiv \mathbb{E}\{x\}$ and covariance $P \equiv \mathbb{E}\{(x-\mu)(x-\mu)'\}$ are given by:

$$\mu = J^{-1}h \quad \text{and} \quad P = J^{-1} \tag{2}$$

The *partial correlation coefficients* are given by:

$$\rho_{i,j} \equiv \frac{\text{cov}(x_i; x_j | x_{V \setminus \{i,j\}})}{\sqrt{\text{var}(x_i | x_{V \setminus \{i,j\}}) \text{var}(x_j | x_{V \setminus \{i,j\}})}} = -\frac{J_{i,j}}{\sqrt{J_{i,i}J_{j,j}}} \tag{3}$$

Thus, $J_{ij} = 0$ if and only if $x_i$ and $x_j$ are independent given the other variables $x_{V \setminus \{i,j\}}$. We say that this model is *attractive* if all partial correlations are non-negative. It is *pairwise-normalizable* if there exists a diagonal matrix $D \succ 0$ and a collection of non-negative definite matrices $\{J_e \succeq 0, e \in \mathcal{E}\}$, where $(J_e)_{i,j}$ is zero unless $i, j \in e$, such that:

$$J = D + \sum_{e \in \mathcal{E}} J_e \tag{4}$$

It is *diagonally-dominant* if for all $i \in V : \sum_{j \neq i} |J_{i,j}| < J_{i,i}$. The class of diagonally-dominant models is a strict subset of the class of pairwise-normalizable models [6].

**Gaussian Elimination and Belief Propagation**   Integrating (1) over all possible values of $x_i$ reduces to *Gaussian elimination* (GE) in the information form (see also [7]), i.e.,

$$p(x_{\setminus i}) \equiv \int p(x_{\setminus i}, x_i) dx_i \propto \exp\{-\frac{1}{2}x'_{\setminus i}\hat{J}_{\setminus i}x_{\setminus i} + \hat{h}'_{\setminus i}x_{\setminus i}\} \tag{5}$$

where $\setminus i \equiv V \setminus \{i\}$, i.e. all variables except $i$, and

$$\hat{J}_{\setminus i} = J_{\setminus i, \setminus i} - J_{\setminus i, i}J_{i,i}^{-1}J_{i,\setminus i} \quad \text{and} \quad \hat{h}_{\setminus i} = h_{\setminus i} - J_{\setminus i, i}J_{i,i}^{-1}h_i \tag{6}$$

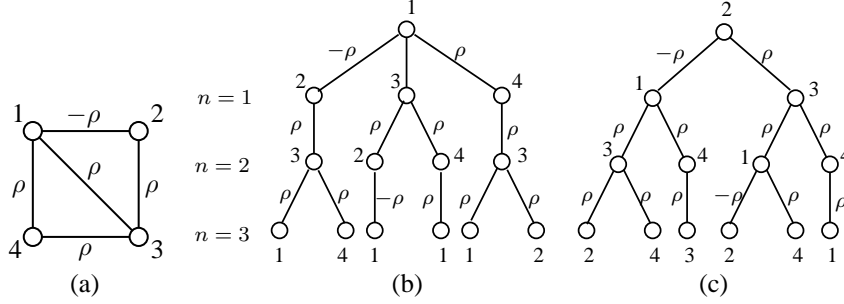

Figure 1: (a) Graph of a GMM with nodes $\{1, 2, 3, 4\}$ and with edge weights (partial correlations) as shown. In (b) and (c) we illustrate the first three levels of the LBP computation tree rooted at nodes 1 and 2. After 3 iterations of LBP in (a), the marginals at nodes 1 and 2 are identical to the marginals at the root of (b) and (c) respectively.

In trees, the marginal of any given node can be efficiently computed by sequentially eliminating leaves of the tree until just that node remains. BP may be seen as a message-passing form of GE in which a message passed from node $i$ to node $j \in N(i)$ captures the effect of eliminating the subtree rooted at $i$. Thus, by a two-pass procedure, BP efficiently computes the marginals at *all* nodes of the tree. The equations for LBP are identical except that messages are updated iteratively and in parallel. There are two messages per edge, one for each ordered pair $(i, j) \in \mathcal{E}$. We specify each message in information form with parameters: $\Delta h_{i \to j}^{(n)}, \Delta J_{i \to j}^{(n)}$ (initialized to zero for $n = 0$). These are iteratively updated as follows. For each $(i, j) \in \mathcal{E}$, messages from $N(i) \setminus j$ are fused at node $i$:

$$\hat{h}_{i \setminus j}^{(n)} = h_i + \sum_{k \in N(i) \setminus j} \Delta h_{k \to i}^{(n)} \quad \text{and} \quad \hat{J}_{i \setminus j}^{(n)} = J_{i,i} + \sum_{k \in N(i) \setminus j} \Delta J_{k \to i}^{(n)} \tag{7}$$

This fused information at node $i$ is predicted to node $j$:

$$\Delta h_{i \to j}^{(n+1)} = -J_{j,i} (\hat{J}_{i \setminus j}^{(n)})^{-1} \hat{h}_{i \setminus j}^{(n)} \quad \text{and} \quad \Delta J_{i \to j}^{(n+1)} = -J_{j,i} (\hat{J}_{i \setminus j}^{(n)})^{-1} J_{i,j} \tag{8}$$

After $n$ iterations, the marginal of node $i$ is obtained by fusing all incoming messages:

$$\hat{h}_i^{(n)} = h_i + \sum_{k \in N(i)} \Delta h_{k \to i}^{(n)} \quad \text{and} \quad \hat{J}_i^{(n)} = J_{i,i} + \sum_{k \in N(i)} \Delta J_{k \to i}^{(n)} \tag{9}$$

The mean and variance are given by $(\hat{J}_i^{(n)})^{-1} \hat{h}_i^{(n)}$ and $(\hat{J}_i^{(n)})^{-1}$. In trees, this is the marginal at node $i$ conditioned on zero boundary conditions at nodes $(n + 1)$ steps away and LBP converges to the correct marginals after a finite number of steps equal to the diameter of the tree. In graphs with cycles, LBP may not converge and only yields approximate marginals when it does. A useful fact about LBP is the following [2, 3, 5]: the marginal computed at node $i$ after $n$ iterations is identical to the marginal at the root of the $n$-step *computation tree* rooted at node $i$. This tree is obtained by "unwinding" the loopy graph for $n$ steps (see Fig. 1). Note that each node of the graph may be replicated many times in the computation tree. Also, neighbors of a node in the computation tree correspond exactly with neighbors of the associated node in the original graph (except at the last level of the tree where some neighbors are missing). The corresponding $J$ matrix defined on the computation tree has the same node and edge values as in the original GMM.

## 3   Walk-Summable Gauss-Markov Models

In this section we present the walk-sum formulation of inference in GMMs. Let $\varrho(A)$ denote the *spectral radius* of a symmetric matrix $A$, defined to be the maximum of the absolute values of the eigenvalues of $A$. The geometric series $(I + A + A^2 + \dots)$ converges

if and only if $\varrho(A) < 1$. If it converges, it converges to $(I - A)^{-1}$. Now, consider a GMM with information matrix $J$. Without loss of generality, let $J$ be normalized (by rescaling variables) to have $J_{i,i} = 1$ for all $i$. Then, $\rho_{i,j} = -J_{i,j}$ and the (zero-diagonal) matrix of partial correlations is given by $R = I - J$. If $\varrho(R) < 1$, then we have a geometric series for the covariance matrix:

$$\sum_{l=0}^{\infty} R^l = (I - R)^{-1} = J^{-1} = P \tag{10}$$

Let $\bar{R} = (|r_{ij}|)$ denote the matrix of element-wise absolute values. We say that the model is *walk-summable* if $\varrho(\bar{R}) < 1$. Walk-summability implies $\varrho(R) < 1$ and $J \succ 0$.

*Example 1.* Consider a 5-node cycle with normalized information matrix $J$, which has all partial correlations on the edges set to $\rho$. If $\rho = -.45$, then the model is valid (i.e. positive definite) with minimum eigenvalue $\lambda_{\min}(J) \approx .2719 > 0$, and walk-summable with $\varrho(\bar{R}) = .9 < 1$. However, when $\rho = -.55$, then the model is still valid with $\lambda_{\min}(J) \approx .1101 > 0$, but no longer walk-summable with $\varrho(\bar{R}) = 1.1 > 1$.

Walk-summability allows us to interpret (10) as computing walk-sums in the graph. Recall that the matrix $R$ reflects graph structure: $\rho_{i,j} = 0$ if $\{i, j\} \notin \mathcal{E}$. These act as weights on the edges of the graph. A *walk* $w = (w_0, w_1, ..., w_l)$ is a sequence of nodes $w_i \in V$ connected by edges $\{w_i, w_{i+1}\} \in \mathcal{E}$ where $l$ is the *length* of the walk. The *weight* $\rho(w)$ of walk $w$ is the product of edge weights along the walk:

$$\rho(w) = \prod_{s=1}^{l} \rho_{w_{s-1}, w_s} \tag{11}$$

At each node $i \in V$, we also define a zero-length walk $w = (i)$ for which $\rho(w) = 1$.

*Walk-Sums.* Given a set of walks $\mathcal{W}$, we define the *walk-sum* over $\mathcal{W}$ by

$$\rho(\mathcal{W}) = \sum_{w \in \mathcal{W}} \rho(w) \tag{12}$$

which is well-defined (i.e., independent of summation order) because $\varrho(\bar{R}) < 1$ implies absolute convergence. Let $\mathcal{W}_{i \overset{l}{\to} j}$ denote the set of $l$-length walks from $i$ to $j$ and let $\mathcal{W}_{i \to j} = \cup_{l=0}^{\infty} \mathcal{W}_{i \overset{l}{\to} j}$. The relation between walks and the geometric series (10) is that the entries of $R^l$ correspond to walk-sums over $l$-length walks from $i$ to $j$ in the graph, i.e., $(R^l)_{i,j} = \rho(\mathcal{W}_{i \overset{l}{\to} j})$. Hence,

$$P_{i,j} = \sum_{l=0}^{\infty} (R^l)_{i,j} = \sum_{l} \rho(\mathcal{W}_{i \overset{l}{\to} j}) = \rho(\cup_l \mathcal{W}_{i \overset{l}{\to} j}) = \rho(\mathcal{W}_{i \to j}) \tag{13}$$

In particular, the variance $\sigma_i^2 \equiv P_{i,i}$ of variable $i$ is the walk-sum taken over the set $\mathcal{W}_{i \to i}$ of *self-return walks* that begin and end at $i$ (defined so that $(i) \in \mathcal{W}_{i \to i}$). The means can be computed as reweighted walk-sums, i.e., where each walk is scaled by the potential at the start of the walk: $\rho(w; h) = h_{w_0} \rho(w)$, and $\rho(\mathcal{W}; h) = \sum_{w \in \mathcal{W}} \rho(w; h)$. Then,

$$\mu_i = \sum_{j \in V} P_{i,j} h_j = \sum_{j} \rho(\mathcal{W}_{j \to i}) h_j = \rho(\mathcal{W}_{* \to i}; h) \tag{14}$$

where $\mathcal{W}_{* \to i} \equiv \cup_{j \in V} \mathcal{W}_{j \to i}$ is the set of all walks which end at node $i$.

We have found that a wide class of GMMs are walk-summable:

**Proposition 1 (Walk-Summable GMMs)** *All of the following classes of GMMs are walk-summable:[2] (i) attractive models, (ii) trees and (iii) pairwise-normalizable[3] models.*

*Proof Outline.* (i) $R = \bar{R}$ and $J = I - \bar{R} \succ 0$ implies $\lambda_{\max}(\bar{R}) < 1$. Because $\bar{R}$ has non-negative elements, $\varrho(\bar{R}) = \lambda_{\max}(\bar{R}) < 1$. In (ii) & (iii), negating any $\rho_{ij}$, it still holds that $J = I - R \succ 0$ : (ii) negating $\rho_{ij}$ doesn't affect the eigenvalues of $J$ (remove edge $\{i, j\}$ and, in each eigenvector, negate all entries in one subtree); (iii) negating $\rho_{ij}$ preserves $J_{\{i,j\}} \succeq 0$ in (4) so $J \succ 0$. Thus, making all $\rho_{ij} > 0$, we find $I - \bar{R} \succ 0$ and $\bar{R} \prec I$. Similarly, making all $\rho_{ij} < 0$, $-\bar{R} \prec I$. Therefore, $\varrho(\bar{R}) < 1$. $\diamond$

## 4 Recursive Walk-Sum Calculations on Trees

In this section we derive a recursive algorithm which accrues the walk-sums (over infinite sets of walks) necessary for exact inference on trees and relate this to BP. Walk-summability guarantees correctness of this algorithm which reorders walks in a non-trivial way.

We start with a chain of $N$ nodes: its graph $\mathcal{G}$ has nodes $V = \{1, \ldots, N\}$ and edges $\mathcal{E} = \{e_1, .., e_{N-1}\}$ where $e_i = \{i, i+1\}$. The variance at node $i$ is $\sigma_i^2 = \rho(\mathcal{W}_{i \to i})$. The set $\mathcal{W}_{i \to i}$ can be partitioned according to the number of times that walks return to node $i$: $\mathcal{W}_{i \to i} = \cup_{r=0}^{\infty} \mathcal{W}_{i \to i}^{(r)}$ where $\mathcal{W}_{i \to i}^{(r)}$ is the set of all self-return walks which return to $i$ exactly $r$ times. In particular, $\mathcal{W}_{i \to i}^{(0)} = \{(i)\}$ for which $\rho(\mathcal{W}_{i \to i}^{(0)}) = 1$. A walk which starts at node $i$ and returns $r$ times is a concatenation of $r$ single-revisit self-return walks, so $\rho(\mathcal{W}_{i \to i}^{(r)}) = \rho(\mathcal{W}_{i \to i}^{(1)})^r$. This means:

$$\rho(\mathcal{W}_{i \to i}) = \rho(\cup_{r=0}^{\infty} \mathcal{W}_{i \to i}^{(r)}) = \sum_{r=0}^{\infty} \rho(\mathcal{W}_{i \to i}^{(r)}) = \sum_{r=0}^{\infty} \rho(\mathcal{W}_{i \to i}^{(1)})^r = \frac{1}{1 - \rho(\mathcal{W}_{i \to i}^{(1)})} \quad (15)$$

This geometric series converges since the model is walk-summable. Hence, calculating the single-revisit self-return walk-sum $\rho(\mathcal{W}_{i \to i}^{(1)})$ determines the variance $\sigma_i^2$. The single-revisit walks at node $i$ consist of walks in the left subchain, and walks in the right subchain. Let $\mathcal{W}_{i \to i \setminus j}$ be the set of self-return walks of $i$ which never visit $j$, so e.g. all $w \in \mathcal{W}_{i \to i \setminus i+1}$ are contained in the subgraph $\{1, \ldots, i\}$. With this notation:

$$\rho(\mathcal{W}_{i \to i}^{(1)}) = \rho(\mathcal{W}_{i \to i \setminus i+1}^{(1)}) + \rho(\mathcal{W}_{i \to i \setminus i-1}^{(1)}) \quad (16)$$

The left single-revisit self-return walk-sums $\rho(\mathcal{W}_{i \to i \setminus i+1}^{(1)})$ can be computed recursively starting from node 1. At node 1, $\rho(\mathcal{W}_{1 \to 1 \setminus 2}^{(1)}) = 0$ and $\rho(\mathcal{W}_{1 \to 1 \setminus 2}) = 1$. A single-revisit self-return walk from node $i$ in the left subchain consists of a step to node $i-1$, then some number of self-return walks in the subgraph $\{1, \ldots, i-1\}$, and a step from $i-1$ back to $i$:

$$\rho(\mathcal{W}_{i \to i \setminus i+1}^{(1)}) = \rho_{i,i-1}^2 \rho(\mathcal{W}_{i-1 \to i-1 \setminus i}) = \frac{\rho_{i,i-1}^2}{1 - \rho(\mathcal{W}_{i-1 \to i-1 \setminus i}^{(1)})} \quad (17)$$

Thus single-revisit (and multiple revisit) walk-sums in the left subchain of every node $i$ can be calculated in one forward pass through the chain. The same can be done for the right subchain walk-sums at every node $i$, by starting at node $N$, and going backwards. Using equations (15) and (16) these quantities suffice to calculate the variances at *all* nodes of the chain. A similar forwards-backwards procedure computes the means as reweighted walk-sums over the left and right single-visit walks for node $i$, which start at an arbitrary node (in the left or right subchain) and end at $i$, never visiting $i$ before that [6]. In fact, these recursive walk-sum calculations map exactly to operations in BP – e.g., in a normalized chain $\Delta J_{i-1 \to i} = -\rho(\mathcal{W}_{i \to i \setminus i+1}^{(1)})$ and $\Delta h_{i-1 \to i} = -\rho(\mathcal{W}_{* \to i \setminus i+1}^{(1)}; h)$. The same strategy applies for trees: both single-revisit and single-visit walks at node $i$ can be partitioned according to which subtree (rooted at a neighbor $j \in N(i)$ of $i$) the walk lives in. This leads to a two-pass walks-sum calculation on trees (from the leaves to the root, and back) to calculate means and variances at all nodes.

# 5 Walk-sum Analysis of Loopy Belief Propagation

First, we analyze LBP in the case that the model is walk-summable and show that LBP converges and includes all the walks for the means, but only a subset of the walks for the variances. Then, we consider the case of non-walksummable models and relate convergence of the LBP variances to walk-summability of the computation tree.

## 5.1 LBP in walk-summable models

To compute means and variances in a walk-summable model, we need to calculate walk-sums for certain sets of walks in the graph $\mathcal{G}$. Running LBP in $\mathcal{G}$ is equivalent to exact inference in the computation tree for $\mathcal{G}$, and hence calculating walk-sums for certain walks in the computation tree. In the computation tree rooted at node $i$, walks ending at the root have a one-to-one correspondence with walks ending at node $i$ in $\mathcal{G}$. Hence, LBP captures all of the walks necessary to calculate the means. For variances, the walks captured by LBP have to start and end at the root in the computation tree. However, some of the self-return walks in $\mathcal{G}$ translate to walks in the computation tree that end at the root but start at a replica of the root, rather than at the root itself. These walks are not captured by the LBP variances. For example, in Fig. 1(a), the walk $(1, 2, 3, 1)$ is a self-return walk in the original graph $\mathcal{G}$ but is *not* a self-return walk in the computation tree shown in Fig. 1(b). LBP variances capture only those self-return walks of the original graph $\mathcal{G}$ which also are self-return walks in the computation tree – e.g., the walk $(1, 3, 2, 3, 4, 3, 1)$ is a self-return walk in both Figs. 1(a) and (b). We call these *backtracking walks*. These simple observations lead to our main result:

**Proposition 2 (Convergence of LBP for walk-summable GMMs)** *If the model is walk-summable, then LBP converges: the means converge to the true means and the LBP variances converge to walk-sums over just the backtracking self-return walks at each node.*

*Proof Outline*. All backtracking walks have positive weights, since each edge is traversed an even number of times. For a walk-summable model, LBP variances are walks-sums over the backtracking walks and are therefore monotonically increasing with the iterations. They also are bounded above by the absolute self-return walk-sums (diagonal elements of $\sum_l \bar{R}^l$) and hence converge. For the means: the series $\sum_{l=0}^{\infty} R^l h$ converges absolutely since $|R^l h| \leq \bar{R}^l |h|$, and the series $\sum_l \bar{R}^l |h|$ is a linear combination of terms of the absolutely convergent series $\sum_l \bar{R}^l$. The LBP means are a rearrangement of the absolutely convergent series $\sum_{l=0}^{\infty} R^l h$, so they converge to the same values. $\diamond$

As a corollary, LBP converges for all of the model classes listed in Proposition 1. Also, in attractive models, the LBP variances are less than or equal to the true variances. Correctness of the means was also shown in [3] for pairwise-normalizable models.[4] They also show that LBP variances omit some terms needed for the correct variances. These terms correspond to correlations between the root and its replicas in the computation tree. In our framework, each such correlation is a walk-sum over the subset of non-backtracking self-return walks in $\mathcal{G}$ which, in the computation tree, begin at a particular replica of the root.

*Example 2*. Consider the graph in Fig. 1(a). For $\rho = .39$, the model is walk-summable with $\varrho(\bar{R}) \approx .9990$. For $\rho = .395$ and $\rho = .4$, the model is still valid but is not walk-summable, with $\varrho(\bar{R}) \approx 1.0118$ and $1.0246$ respectively. In Fig. 2(a) we show LBP variances for node 1 (the other nodes are similar) vs. the iteration number. As $\rho$ increases, first the model is walk-summable and LBP converges, then for a small interval the model is not walk-summable but LBP still converges,[5] and for larger $\rho$ LBP does not converge. Also,

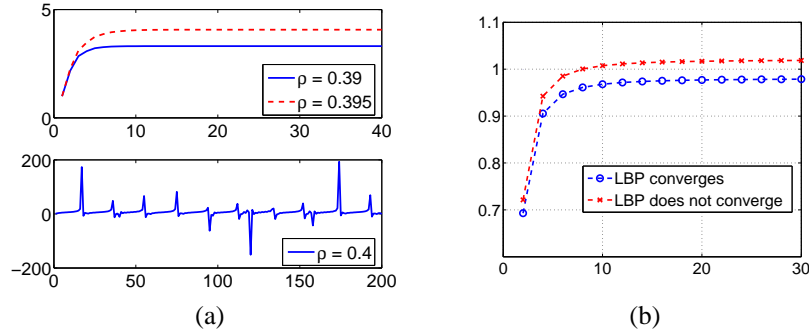

Figure 2: (a) LBP variances vs. iteration. (b) $\varrho(R_n)$ vs. iteration.

for $\rho = .4$, we note that $\varrho(R) = .8 < 1$ and the series $\sum_l R^l$ converges (but $\sum_l \bar{R}^l$ does not) and LBP does not converge. Hence, $\varrho(R) < 1$ is *not* sufficient for LBP convergence showing the importance of the stricter walk-summability condition $\varrho(\bar{R}) < 1$.

## 5.2 LBP in non-walksummable models

We extend our analysis to develop a tighter condition for convergence of LBP variances based on walk-summability of the computation tree (rather than walk-summability on $\mathcal{G}$).[6] For trees, walk-summability and validity are equivalent, i.e. $J \succ 0 \Leftrightarrow \varrho(\bar{R}) < 1$, hence our condition is equivalent to validity of the computation tree.

First, we note that when a model on $\mathcal{G}$ is valid ($J$ is positive-definite) but not walk-summable, then some finite computation trees may be invalid (indefinite). This turns out to be the reason why LBP variances can fail to converge. Walk-summability of the original GMM implies walk-summability (and hence validity) of all of its computation trees. But if the GMM is not walk-summable, then its computation tree may or may not be walk-summable. In Example 2, for $\rho = .395$ the computation tree is still walk-summable (even though the model on $\mathcal{G}$ is not) and LBP converges. For $\rho = .4$, the computation tree is not walk-summable and LBP does not converge. Indeed, LBP is not even well-posed in this case (because the computation tree is indefinite) which explains its strange behavior seen in the bottom plot of Fig. 2(a) (e.g., non-monotonicity and negative variances).

We characterize walk-summability of the computation tree as follows. Let $T_n$ be the $n$-step computation tree rooted at some node $i$ and define $R_n \triangleq I_n - J_n$ where $J_n$ is the normalized information matrix on $T_n$ and $I_n$ is the $n \times n$ identity matrix. The $n$-step computation tree $T_n$ is walk-summable (valid) if and only if $\varrho(R_n) < 1$ (in trees, $\varrho(\bar{R}_n) = \varrho(R_n)$). The sequence $\{\varrho(R_n)\}$ is monotonically increasing and bounded above by $\varrho(\bar{R})$ (see [6]) and hence converges. We are interested in the quantity $\varrho_\infty \equiv \lim_{n \to \infty} \varrho(R_n)$.

**Proposition 3 (LBP validity/variance convergence)** *(i) If $\varrho_\infty < 1$, then all finite computation trees are valid and the LBP variances converge. (ii) If $\varrho_\infty > 1$, then the computation tree eventually becomes invalid and LBP is ill-posed.*

*Proof Outline.* (i) For some $\delta > 0$, $\varrho(R_n) \leq 1 - \delta$ for all $n$ which implies: all computation trees are walk-summable and variances monotonically increase; $\lambda_{\max}(R_n) \leq 1 - \delta$, $\lambda_{\min}(J_n) \geq \delta$, and $(P_n)_{i,i} \leq \lambda_{\max}(P_n) \leq \frac{1}{\delta}$. The variances are monotonically increasing

and bounded above, hence they converge. (ii) If $\lim_{n\to\infty} \varrho(R_n) > 1$, then there exists an $m$ for which $\varrho(R_n) > 1$ for all $n \geq m$ and the computation tree is invalid. $\diamond$

As discussed in [6], LBP is well-posed if and only if the information numbers computed on the right in (7) and (9) are strictly positive for all $n$. Hence, it is easily detected if the LBP computation tree becomes invalid. In this case, continuing to run LBP is not meaningful and will lead to division by zero and/or negative variances.

*Example 3*. Consider a 4-node cycle with edge weights $(-\rho, \rho, \rho, \rho)$. In Fig. 2(b), for $\rho = .49$ we plot $\varrho(R_n)$ vs. $n$ (lower curve) and observe that $\lim_{n\to\infty} \varrho(R_n) \approx .98 < 1$, and LBP converges (similar to the upper plot of Fig. 2(a)). For $\rho = .51$ (upper curve), the model defined on the 4-node cycle is still valid but $\lim_{n\to\infty} \varrho(R_n) \approx 1.02 > 1$ so LBP is ill-posed and does not converge (similar to the lower plot of Fig. 2(a)).

In non-walksummable models, the series LBP computes for the means is not absolutely convergent and may diverge even when variances converge (e.g., in Example 2 with $\rho = .39867$). However, in all cases where variances converge we have observed that with enough damping of BP messages[7] we also obtain convergence of the means. Apparently, it is the validity of the computation tree that is critical for convergence of Gaussian LBP.

## 6  Conclusion

We have presented a walk-sum interpretation of inference in GMMs and have applied this framework to analyze convergence of LBP extending previous results. In future work, we plan to develop extended walk-sum algorithms which gather more walks than LBP. Another approach is to estimate variances by sampling random walks in the graph. We also are interested to explore possible connections between results in [8] – based on self-avoiding walks in Ising models – and sufficient conditions for convergence of discrete LBP [9] which have some parallels to our walk-sum analysis in the Gaussian case.

**Acknowledgments**  This research was supported by the Air Force Office of Scientific Research under Grant FA9550-04-1, the Army Research Office under Grant W911NF-05-1-0207 and by a grant from MIT Lincoln Laboratory.

## Footnotes

[1]The work also applies to $p(x|y)$, i.e. where some variables $y$ are observed. However, the observations $y$ are fixed, and we redefine $p(x) \triangleq p(x|y)$ (conditioning on $y$ is implicit throughout). With local observations $p(x|y) \propto p(x) \prod_i p(y_i|x_i)$, conditioning does not change the graph structure.

[2]That is if we take a valid model (with $J \succ 0$) in these classes then it automatically has $\varrho(\bar{R}) < 1$.

[3]In [6], we also show that walk-summability is actually equivalent to pairwise-normalizability.

[4]However, they only prove convergence for the subset of diagonally dominant models.

[5]Hence, walk-summability is sufficient but not necessary for convergence of LBP.

[6]We can focus on one tree: if the computation tree rooted at node $i$ is walk-summable, then so is the computation tree rooted at any node $j$. Also, if a finite computation tree rooted at node $i$ is not walk-summable, then some finite tree at node $j$ also becomes non-walksummable for $n$ large enough.

[7]Modify (8) as follows: $\Delta h_{i\to j}^{(n+1)} = (1-\alpha)\Delta h_{i\to j}^{(n)} + \alpha(-J_{i,j}(\hat{J}_{i\backslash j}^{(n)})^{-1}\hat{h}_{i\backslash j}^{(n)})$ with $0 < \alpha \leq 1$. In Example 2, with $\rho = .39867$ and $\alpha = .9$ the means converge.

## References

[1] J. Pearl. *Probabilistic inference in intelligent systems*. Morgan Kaufmann, 1988.

[2] J. Yedidia, W. Freeman, and Y. Weiss. Understanding belief propagation and its generalizations. *Exploring AI in the new millennium*, pages 239–269, 2003.

[3] Y. Weiss and W. Freeman. Correctness of belief propagation in Gaussian graphical models of arbitrary topology. *Neural Computation*, 13:2173–2200, 2001.

[4] P. Rusmevichientong and B. Van Roy. An analysis of belief propagation on the turbo decoding graph with Gaussian densities. *IEEE Trans. Information Theory*, 48(2):745–765, Feb. 2001.

[5] S. Tatikonda and M. Jordan. Loopy belief propagation and Gibbs measures. *UAI*, 2002.

[6] J. Johnson, D. Malioutov, and A. Willsky. Walk-Summable Gaussian Networks and Walk-Sum Interpretation of Gaussian Belief Propagation. TR-2650, LIDS, MIT, 2005.

[7] K. Plarre and P. Kumar. Extended message passing algorithm for inference in loopy Gaussian graphical models. *Ad Hoc Networks*, 2004.

[8] M. Fisher. Critical temperatures of anisotropic Ising lattices II, general upper bounds. *Physical Review*, 162(2), 1967.

[9] A. Ihler, J. Fisher III, and A. Willsky. Message Errors in Belief Propagation. *NIPS*, 2004.

